# Theory-Based Causal Inference

**Joshua B. Tenenbaum & Thomas L. Griffiths**
Department of Brain and Cognitive Sciences
MIT, Cambridge, MA 02139
{jbt, gruffydd}@mit.edu

## Abstract

People routinely make sophisticated causal inferences unconsciously, effortlessly, and from very little data – often from just one or a few observations. We argue that these inferences can be explained as Bayesian computations over a hypothesis space of causal graphical models, shaped by strong top-down prior knowledge in the form of intuitive theories. We present two case studies of our approach, including quantitative models of human causal judgments and brief comparisons with traditional bottom-up models of inference.

## 1 Introduction

People are remarkably good at inferring the causal structure of a system from observations of its behavior. Like any inductive task, causal inference is an ill-posed problem: the data we see typically underdetermine the true causal structure. This problem is worse than the usual statistician's dilemma that "correlation does not imply causation". Many cases of everyday causal inference follow from just one or a few observations, where there isn't even enough data to reliably infer correlations! This fact notwithstanding, most conventional accounts of causal inference attempt to generate hypotheses in a bottom-up fashion based on empirical correlations. These include associationist models [12], as well as more recent rational models that embody an explicit concept of causation [1,3], and most algorithms for learning causal Bayes nets [10,14,7].

Here we argue for an alternative top-down approach, within the causal Bayes net framework. In contrast to standard bottom-up approaches to structure learning [10,14,7], which aim to optimize or integrate over all possible causal models (structures and parameters), we propose that people consider only a relatively constrained set of hypotheses determined by their prior knowledge of how the world works. The allowed causal hypotheses not only form a small set of all possible causal graphs, but also instantiate specific causal mechanisms with constrained conditional probability tables, rather than much more general conditional dependence and independence relations.

The prior knowledge that generates this hypothesis space of possible causal models can be thought of as an intuitive theory, analogous to the scientific theories of classical mechanics or electrodynamics that generate constrained spaces of possible causal models in their domains. Following the suggestions of recent work in cognitive development (reviewed in [4]), we take the existence of strong intuitive theories to be the foundation for human causal inference. However, our view contrasts with some recent suggestions [4,11] that

an intuitive theory may be represented as a causal Bayes net model. Rather, we consider a theory to be the underlying principles that generate the range of causal network models potentially applicable in a given domain – the abstractions that allow a learner to construct and reason with appropriate causal network hypotheses about novel systems in the presence of minimal perceptual input.

Given the hypothesis space generated by an intuitive theory, causal inference then follows the standard Bayesian paradigm: weighing each hypothesis according to its posterior probability and averaging their predictions about the system according to those weights. The combination of Bayesian causal inference with strong top-down knowledge is quite powerful, allowing us to explain people's very rapid inferences about model complexity in both static and temporally extended domains. Here we present two case studies of our approach, including quantitative models of human causal judgments and brief comparisons with more bottom-up accounts.

## 2 Inferring hidden causal powers

We begin with a paradigm introduced by Gopnik and Sobel for studying causal inference in children [5]. Subjects are shown a number of blocks, along with a machine – the "blicket detector". The blicket detector "activates" – lights up and makes noise – whenever a "blicket" is placed on it. Some of the blocks are "blickets", others are not, but their outward appearance is no guide. Subjects observe a series of trials, on each of which one or more blocks are placed on the detector and the detector activates or not. They are then asked which blocks have the hidden causal power to activate the machine.

Gopnik and Sobel have demonstrated various conditions under which children successfully infer the causal status of blocks from just one or a few observations. Of particular interest is their "backwards blocking" condition [13]: on trial 1 (the "1-2" trial), children observe two blocks ($B_1$ and $B_2$) placed on the detector and the detector activates. Most children now say that both $B_1$ and $B_2$ are blickets. On trial 2 (the "1 alone" trial), $B_1$ is placed on the detector alone and the detector activates. Now all children say that $B_1$ is a blicket, and most say that $B_2$ is *not* a blicket. Intuitively, this is a kind of "explaining away": seeing that $B_1$ is sufficient to activate the detector alone explains away the previously observed association of $B_2$ with detector activation.

Gopnik et al. [6] suggest that children's causal reasoning here may be thought of in terms of learning the structure of a causal Bayes net. Figure 1a shows a Bayes net, $h_{10}$, that is consistent with children's judgments after trial 2. Variables $X_1$ and $X_2$ represent whether blocks $B_1$ and $B_2$ are on the detector; $E$ represents whether the detector activates; the existence of an edge $X_1 \rightarrow E$ but no edge $X_2 \rightarrow E$ represents the hypothesis that $B_1$ but not $B_2$ is a blicket – that $B_1$ but not $B_2$ has the power to turn on the detector. We encode the two observations $\{d_1, d_2\}$ as vectors $[x_1, x_2, e]$, where $x_1 = 1$ if block 1 is on the detector (else 0), likewise for $x_2$ and block 2, and $e = 1$ if the detector is active (else 0).

Given only the data $\{d_1 = [1, 1, 1], d_2 = [1, 0, 1]\}$, standard Bayes net learning algorithms have no way to converge on subjects's choice $h_{10}$. The data are not sufficient to compute the conditional independence relations required by constraint-based methods [9,13], [1] nor to strongly influence the Bayesian structural score using arbitrary conditional probability tables [7]. Standard psychological models of causal strength judgment [12,3], equivalent to maximum-likelihood parameter estimates for the family of Bayes nets in Figure 1a [15], either predict no explaining away here or make no prediction due to insufficient data.

Alternatively, reasoning on this task could be explained in terms of a simple logical deduction. We require as a premise the *activation law*: a blicket detector activates if and only if one or more blickets are placed on it. Based on the activation law and the data $\{d_1, d_2\}$, we can deduce that $B_1$ is a blicket but $B_2$ remains undetermined. If we further assume a form of Occam's razor, positing the minimal number of hidden causal powers, then we can infer that $B_2$ is not a blicket, as most children do. Other cases studied by Gopnik et al. can be explained similarly. However, this deductive model cannot explain many plausible but nondemonstrative causal inferences that people make, or people's degrees of confidence in their judgments, or their ability to infer probabilistic causal relationships from noisy data [3,12,15]. It also leaves mysterious the origin and form of Occam's razor. In sum, neither deductive logic nor standard Bayes net learning provides a satisfying account of people's rapid causal inferences. We now show how a Bayesian structural inference based on strong top-down knowledge can explain the blicket detector judgments, as well as several probabilistic variants that clearly exceed the capacity of deductive accounts.

Most generally, the top-down knowledge takes the form of a causal theory with at least two components: an ontology of object, attribute and event types, and a set of causal principles relating these elements. Here we treat theories only informally; we are currently developing a formal treatment using the tools of probabilistic relational logic (e.g., [9]). In the basic blicket detector domain, we have two kinds of objects, blocks and machines; two relevant attributes, being a blicket and being a blicket detector; and two kinds of events, a block being placed on a machine and a machine activating. The causal principle relating these events and attributes is just the activation law introduced above. Instead of serving as a premise for deductive inference, the causal law now generates a hypothesis space of causal Bayes nets for statistical inference. This space is quite restricted: with two objects and one detector, there are only 4 consistent hypotheses $\mathcal{H} = \{h_{10}, h_{01}, h_{11}, h_{00}\}$ (Figure 1a). The conditional probabilities for each hypothesis $h_{jk}$ are also determined by the theory. Based on the activation law, $p(e|x_1, x_2, h_{jk}) = 1$ if $j = 1$ and $x_1 = 1$, or $k = 1$ and $x_2 = 1$; otherwise it equals 0.

Causal inference then follows by Bayesian updating of probabilities over $\mathcal{H}$ in light of the observed data $\mathcal{D}$. We assume independent observations so that the total likelihood factors into separate terms for individual trials. For all hypotheses in $\mathcal{H}$, the individual-trial likelihoods also factor into $p(e|x_1, x_2, h_{jk}) \, p(x_1|h_{jk}) \, p(x_2|h_{jk})$, and we can ignore the last two terms $p(x_1|h_{jk}) \, p(x_2|h_{jk})$ assuming that block positions are independent of the causal structure. The remaining term $p(e|x_1, x_2, h_{jk})$ is 1 for any hypothesis consistent with the data and 0 otherwise, because of the deterministic activation law. The posterior $p(h_{jk}|\mathcal{D})$ for any data set $\mathcal{D}$ is then simply the restriction and renormalization of the prior $p(h_{jk})$ to the set of hypotheses consistent with $\mathcal{D}$. [2]

Backwards blocking proceeds as follows. After the "1-2" trial ($d_1$), at least one block must be a blicket: the consistent hypotheses are $h_{10}$, $h_{01}$, and $h_{11}$. After the "1 alone" trial ($d_2$), only $h_{10}$ and $h_{11}$ remain consistent. The prior over causal structures $p(h_{jk})$ can be written as $\rho^j (1 - \rho)^{1-j} \rho^k (1 - \rho)^{1-k}$, assuming that each block has some independent probability $\rho$ of being a blicket. The nonzero posterior probabilities are then given as follows (all others are zero): $p(h_{10}|d_1) = p(h_{01}|d_1) = \frac{\rho(1-\rho)}{\rho^2 + 2\rho(1-\rho)}$, $p(h_{11}|d_1) = \frac{\rho^2}{\rho^2 + 2\rho(1-\rho)}$, $p(h_{10}|d_1, d_2) = \frac{\rho(1-\rho)}{\rho^2 + \rho(1-\rho)} = 1 - \rho$, and $p(h_{11}|d_1, d_2) = \frac{\rho^2}{\rho^2 + \rho(1-\rho)} = \rho$. Finally, the probability that block $i$ is a blicket $p(X_i \rightarrow E|\mathcal{D})$ may be computed by averaging the predictions of all consistent hypotheses weighted their posterior probabilities: $p(X_1 \rightarrow E|\mathcal{D}) = \sum_{jk} p(X_1 \rightarrow E|h_{jk})p(h_{jk}|\mathcal{D}) = \sum_k (h_{1k}|\mathcal{D})$, $p(X_2 \rightarrow E|\mathcal{D}) = \sum_j p(h_{j1}|\mathcal{D})$.

In comparing with human judgments in the backwards blocking paradigm, the relevant probabilities are $p(X_i \rightarrow E)$, the baseline judgments before either block is placed on the detector; $p(X_i \rightarrow E|d_1)$, judgments after the "1-2" trial; and $p(X_i \rightarrow E|d_1, d_2)$, judgments after the "1 alone" trial. These probabilities depend only on the prior probability of blickets, $\rho$. Setting $\rho = 1/3$ qualitatively matches children's backwards blocking behavior: after the "1-2" trial, both blocks are more likely than not to be blickets ($p(X_i \rightarrow E|d_1) = 3/5$; then, after the "1 alone" trial, $B_1$ is definitely a blicket while $B_2$ is probably not ($p(X_2 \rightarrow E|d_1, d_2) = 1/3$). Thus there is no need to posit a special "Occam's razor" just to explain why $B_2$ becomes like less likely to be a blicket after the "1 alone" trial – this adjustment follows naturally as a rational statistical inference. However, we do have to assume that blickets are somewhat rare ($\rho < 1/2$). Following the "1 alone" trial the probability of $B_2$ being a blicket returns to baseline ($\rho$), because the unambiguous second trial explains away all the evidence for $B_2$ from the first trial. Thus for $\rho > 1/2$, block 2 would remain likely to be a blicket even after the "1 alone" trial.

In order to test whether human causal reasoning actually embodies this Bayesian form of Occam's razor, or instead a more qualitative rule such as the classical version, "Entities should not be multiplied beyond necessity", we conducted three new blicket-detector experiments on both adults and 4-year-old children (in collaboration with Sobel & Gopnik). The first two experiments were just like the original backwards blocking studies, except that we manipulated subjects' estimates of $\rho$ by introducing a pretraining phase. Subjects first saw 12 objects placed on the detector, of which either 2, in the "rare" condition", or 10, in the "common" condition, activated the detector. We hypothesized that this manipulation would lead subjects to set their subjective prior for blickets to either $\rho \approx 1/6$ or $\rho \approx 5/6$, and thus, if guided by the Bayesian Occam's razor, to show strong or weak blocking respectively.

We gave adult subjects a different cover story, involving "super pencils" and a "superlead detector", but here we translate the results into blicket detector terms. Following the "rare" or "common" training, two new objects $B_1$ and $B_2$ were picked at random from the same pile and subjects were asked three times to judge the probability that each one could activate the detector: first, before seeing it on the detector, as a baseline; second, after a "1-2" trial; third, after a "1 alone" trial. Probabilities were judged on a 1-7 scale and then rescaled to the range 0-1.

The mean adult probability judgments and the model predictions are shown in Figures 2a (rare) and 2b (common). Wherever two objects have the same pattern of observed contingencies (e.g., $B_1$ and $B_2$ at baseline and after the "1-2" trial), subjects' mean judgments were found not to be significantly different and were averaged together for this analysis. In fitting the model, we adjusted $\rho$ to match subjects' baseline judgments; the best-fitting values were very close to the true base rates. More interestingly, subjects' judgments tracked the Bayesian model over both trials and conditions. Following the "1-2" trial, mean ratings of both objects increased above baseline, but more so in the rare condition where the activation of the detector was more surprising. Following the "1 alone" trial, all subjects in both conditions were 100% sure that $B_1$ had the power to activate the detector, and the mean rating of $B_2$ returned to baseline: low in the rare condition, but high in the common condition. Four-year-old children made "yes"/"no" judgments that were qualitatively similar, across both rare and common conditions [13].

Human causal inference thus appears to follow rational statistical principles, obeying the Bayesian version of Occam's razor rather than the classical logical version. However, an alternative explanation of our data is that subjects are simply employing a combination of logical reasoning and simple heuristics. Following the "1 alone" trial, people could logically deduce that they have no information about the status of $B_2$ and then fall back on the base rate of blickets as a default, without the need for any genuinely Bayesian computations. To rule out this possibility, our third study tested causal explaining way in the

absence of unambiguous data that could be used to support deductive reasoning. Subjects again saw the "rare" pretraining, but now the critical trials involved three objects, $B_1$, $B_2$, $B_3$. After judging the baseline probability that each object could activate the detector, subjects saw two trials: a "1-2" trial, followed by a "1-3" trial, in which objects $B_1$ and $B_3$ activated the detector together. The Bayesian hypothesis space is analogous to Figure 1a, but now includes eight ($2^3$) hypotheses $h_{jkl}$ representing all possible assignments of causal powers to the three objects. As before, the prior over causal structures $p(h_{jkl})$ can be written as $\rho^j(1-\rho)^{1-j}\rho^k(1-\rho)^{1-k}\rho^l(1-\rho)^{1-l}$, the likelihood $p(\mathcal{D}|h_{jkl})$ reduces to 1 for any hypothesis consistent with $\mathcal{D}$ (under the activation law) and 0 otherwise, and the probability that block $i$ is a blicket $p(X_i \to E|\mathcal{D})$ may be computed by summing the posterior probabilities of all consistent hypotheses, e.g., $p(X_1 \to E|\mathcal{D}) = \sum_{kl} p(h_{1kl}|\mathcal{D})$.

Figure 2c shows that the Bayesian model's predictions and subjects' mean judgments match well except for a slight overshoot in the model. Following the "1-3" trial, people judge that $B_1$ probably activates the detector, but now with less than 100% confidence. Correspondingly, the probability that $B_2$ activates the detector decreases, and the probability that $B_3$ activates the detector increases, to a level above baseline but below 0.5. All of these predicted effects are statistically significant ($p < 0.05$, one-tailed paired t-tests).

These results provide strong support for our claim that rapid human inferences about causal structure can be explained as theory-guided Bayesian computations. Particularly striking is the contrast between the effects of the "1 alone" trial and the "1-3 trial". In the former case, subjects observe unambiguously that $B_1$ is a cause and their judgment about $B_2$ falls completely to baseline; in the latter, they observe only a suspicious coincidence and so explaining away is not complete. A logical deductive mechanism might generate the all-or-none explaining-away observed in the former case, while a bottom-up associative learning mechanism might generate the incomplete effect seen in the latter case, but only our top-down Bayesian approach naturally explains the full spectrum of one-shot causal inferences, from uncertainty to certainty.

## 3  Causal inference in perception

Our second case study argues for the importance of causal theories in a very different domain: perceiving the mechanics of collisions and vibrations. Michotte's [8] studies of causal perception showed that a moving ball coming to rest next to a stationary ball would be perceived as the cause of the latter's subsequent motion only if there was essentially no gap in space or time between the end of the first ball's motion and the beginning of the second ball's. The standard explanation is that people have automatic perceptual mechanisms for detecting certain kinds of physical causal relations, such as transfer of force, and these mechanisms are driven by simple bottom-up cues such as spatial and temporal proximity.

Figure 3a shows data from an experiment described in [2] which might appear to support this view. Subjects viewed a computer screen depicting a long horizontal beam. At one end of the beam was a trap door, closed at the beginning of each trial. On each trial, a heavy block was dropped onto the beam at some position $X$, and after some time $T$, the trap door opened and a ball flew out. Subjects were told that the block dropping on the beam might have jarred loose a latch that opens the door, and they were asked to judge (on a $1-7$ scale) how likely it was that the block dropping was the cause of the door opening. The distance $X$ and time $T$ separating these two events were varied across trials. Figure 3a shows that as either $X$ or $T$ increases, the judged probability of a causal link decreases.

Anderson [1] proposed that this judgment could be formalized as a Bayesian inference with two alternative hypotheses: $h_1$, that a causal link exists, and $h_0$, that no causal link exists. He suggested that the likelihood $p(T, X|h_1)$ should be product of decreasing exponentials in space and time, $p(T, X|h_1) = ab \exp(-aT) \exp(-bX)$, while $p(T, X|h_0)$ would pre-

sumably be constant. This model has three free parameters – the decay constants $a$ and $b$, and the prior probability $p(h_1)$ – plus multiplicative and additive scaling parameters to bring the model ouptuts onto the same range as the data. Figure 3c shows that this model can be adjusted to fit the broad outlines of the data, but it misses the crossover interaction: in the data, but not the model, the typical advantage of small distances $X$ over large distances disappears and even reverses as $T$ increases.

This crossover may reflect the presence of a much more sophisticated theory of force transfer than is captured by the spatiotemporal decay model. Figure 1b shows a causal graphical structure representing a simplified physical model of this situation. The graph is a dynamic Bayes net (DBN), enabling inferences about the system's behavior over time. There are four basic event types, each indexed by time $t$. The door state $E(t)$ can be either open ($E(t) = 1$) or closed ($E(t) = 0$), and once open it stays open. There is an intrinsic source of noise $Z(t)$ in the door mechanism, which we take to be i.i.d., zero-mean gaussian. At each time step $t$, the door opens if and only if the noise amplitude $|Z(t)|$ exceeds some threshold (which we take to be 1 without loss of generality). The block hits the beam at position $X(0)$ (and time $t = 0$), setting up a vibration in the door mechanism with energy $V(0)$. We assume this energy decreases according to an inverse power law with the distance between the block and the door, $V(0) = cX(0)^{-\gamma}$. (We can always set $c = 1$, absorbing it into the parameter $\alpha$ below.) For simplicity, we assume that energy propagates instantaneously from the block to the door (plausible given the speed of sound relative to the distances and times used here), and that there is no vibrational damping over time ($V(t) = V(t-1)$). Anderson [2] also sketches an account along these lines, although he provides no formal model.

At time $t = T$, the door pops open; we denote this event as $e(T)$. The likelihood of $e(T)$ depends strictly on the variance of the noise $Z$ – the bigger the variance, the sooner the door should pop open. At issue is whether there exists a causal link between the vibration $V$ – caused by the block dropping – and the noise $Z$ – which causes the door to open. More precisely, we propose that causal inference is based on the probabilities $p(e(T)|X, h_k)$ under the two hypotheses $h_1$ (causal link) and $h_0$ (no causal link). The noise variance has some low intrinsic level $\beta$, which under $h_1$ – but not $h_0$ – is increased by some fraction $\alpha$ of the vibrational energy $V$. That is, $p(Z(t)|V(t), h_k) = \mathcal{N}(Z(t); 0, \beta + k\alpha V(t))$. We can then solve for the likelihoods $p(e(T)|X, h_k)$ analytically or through simulation. We take the limit as the intrinsic noise level $\beta \to 0$, leaving three free parameters, $\alpha$, $\gamma$, and $p(h_1)$, plus multiplicative and additive scaling parameters, just as in the spatiotemporal decay model. Figure 3b plots the (scaled) posterior probabilities $p(h_1|e(T), X)$ for the best fitting parameter values. In contrast to the spatiotemporal decay model, the DBN model captures the crossover interaction between space and time.

This difference between the two models is fundamental, not just an accident of the parameter values chosen. The spatiotemporal decay model can never produce a crossover effect due to its functional form – separable in $T$ and $X$. A crossover of some form is generic in the DBN model, because its predictions essentially follow an exponential decay function on $T$ with a decay rate that is a nonlinear function of $X$. Other mathematical models with a nonseparable form could surely be devised to fit this data as well. The strength of our model lies in its combination of rational statistical inference and realistic physical motivation. These results suggest that whatever schema of force transfer is in people's brains, it must embody a more complex interaction between spatial and temporal factors than is assumed in traditional bottom-up models of causal inference, and its functional form may be a rational consequence of a rich but implicit physical theory that underlies people's instantaneous percepts of causality. It is an interesting open question whether human observers can use this knowledge only by carrying out an online simulation in parallel with their observations, or can access it in a "compiled" form to interpret bottom-up spatiotemporal cues without the need to conduct any explicit internal simulations.

# 4  Conclusion

In two case studies, we have explored how people make rapid inferences about the causal texture of their environment. We have argued that these inferences can be explained best as Bayesian computations, working over hypothesis spaces strongly constrained by top-down causal theories. This framework allowed us to construct quantitative models of causal judgment – the most accurate models to date in both domains, and in the blicket detector domain, the only quantitatively predictive model to date. Our models make a number of substantive and mechanistic assumptions about aspects of the environment that are not directly accessible to human observers. From a scientific standpoint this might seem undesirable; we would like to work towards models that require the fewest number of a priori assumptions. Yet we feel there is no escaping the need for powerful top-down constraints on causal inference, in the form of intuitive theories. In ongoing work, we are beginning to study the origins of these theories themselves. We expect that Bayesian learning mechanisms similar to those considered here will also be useful in understanding how we acquire the ingredients of theories: abstract causal principles and ontological types.

## Footnotes

[1]Gopnik et al. [6] argue that constraint-based learning could be applied here, if we supplement the observed data with large numbers of fictional observations. However, this account does not explain why subjects make the inferences that they do from the very limited data actually observed, nor why they are justified in doing so. Nor does it generalize to the three experiments we present here.

[2]More generally, we could allow for some noise in the detector, by letting the likelihood $p(e|x_1, x_2, h_{jk})$ be probabilistic rather than deterministic. For simplicity we consider only the noiseless case here; a low level of noise would give similar results.

# References

[1] J. .R. Anderson. *The Adaptive Character of Thought*. Erlbaum, 1990.

[2] J. .R. Anderson. Is human cognition adaptive? *Behavioral and Brain Sciences*, 14, 471–484, 1991.

[3] P. W. Cheng. From covariation to causation: A causal power theory. *Psychological Review*, 104, 367–405, 1997.

[4] A. Gopnik & C. Glymour. Causal maps and Bayes nets: a cognitive and computational account of theory-formation. In Carruthers et al. (eds.), *The Cognitive Basis of Science*. Cambridge, 2002.

[5] A. Gopnik & D. M. Sobel. Detecting blickets: How young children use information about causal properties in categorization and induction. *Child Development*, 71, 1205–1222, 2000.

[6] A. Gopnik, C. Glymour, D. M. Sobel, L. E. Schulz, T. Kushnir, D. Danks. A theory of causal learning in children: Causal maps and Bayes nets. *Psychological Review*, in press.

[7] D. Heckerman. A Bayesian approach to learning causal networks. In *Proc. Eleventh Conf. on Uncertainty in Artificial Intelligence*, Morgan Kaufmann Publishers, San Francisco, CA, 1995.

[8] A. E. Michotte. *The Perception of Causality*. Basic Books, 1963.

[9] H. Pasula & S. Russell. Approximate inference for first-order probabilistic languages. In *Proc. International Joint Conference on Artificial Intelligence*, Seattle, 2001.

[10] J. Pearl. *Causality*. New York: Oxford University Press, 2000.

[11] B. Rehder. A causal-model theory of conceptual representation and categorization. Submitted for publication, 2001.

[12] D. R. Shanks. Is human learning rational? *Quarterly Journal of Experimental Psychology*, 48a, 257–279, 1995.

[13] D. Sobel, J. B. Tenenbaum & A. Gopnik. The development of causal learning based on indirect evidence: More than associations. Submitted for publication, 2002.

[14] P. Spirtes, C. Glymour, & R. Scheines. *Causation, prediction, and search (2nd edition, revised)*. Cambridge, MA: MIT Press, 2001.

[15] J. B. Tenenbaum & T. L. Griffiths. Structure learning in human causal induction. In T. Leen, T. Dietterich, and V. Tresp (eds.), *Advances in Neural Information Processing Systems 13*. Cambridge, MA: MIT Press, 2001.

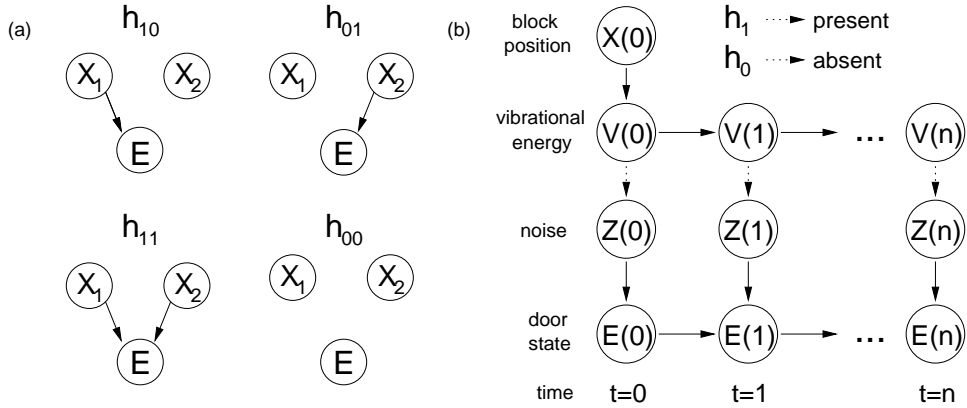

Figure 1: Hypothesis spaces of causal Bayes nets for (a) the blicket detector and (b) the mechanical vibration domains.

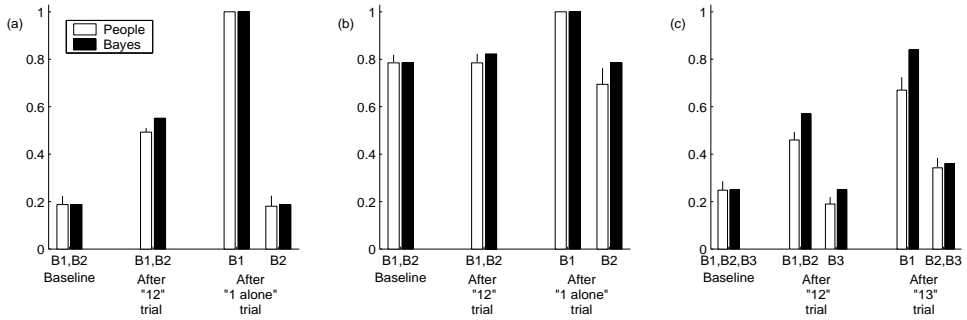

Figure 2: Human judgments and model predictions (based on Figure 1a) for one-shot backwards blocking with blickets, when blickets are (a) rare or (b) common, or (c) rare and only observed in ambiguous combinations. Bar height represents the mean judged probability that an object has the causal power to activate the detector.

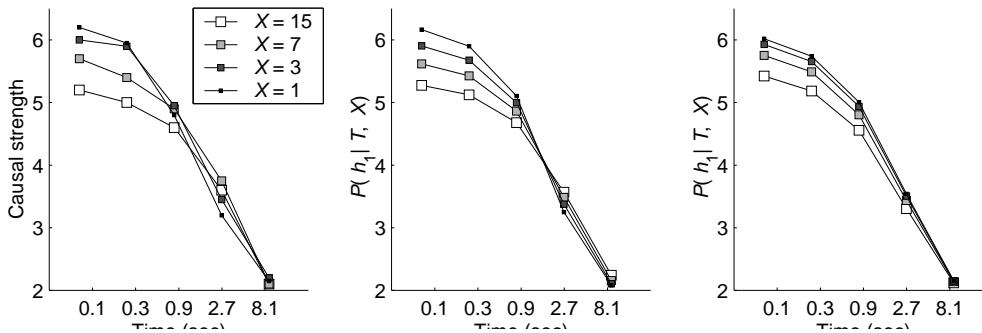

Figure 3: Probability of a causal connection between two events: a block dropping onto a beam and a trap door opening. Each curve corresponds to a different spatial gap $X$ between these events; each x-axis value to a different temporal gap $T$. (a) Human judgments. (b) Predictions of the dynamic Bayes net model (Figure 1b). (c) Predictions of the spatiotemporal decay model.